# Robust Full Bayesian Methods for Neural Networks

**Christophe Andrieu***
Cambridge University
Engineering Department
Cambridge CB2 1PZ
England
ca226@eng.cam.ac.uk

**João FG de Freitas**
UC Berkeley
Computer Science
387 Soda Hall, Berkeley
CA 94720-1776 USA
jfgf@cs.berkeley.edu

**Arnaud Doucet**
Cambridge University
Engineering Department
Cambridge CB2 1PZ
England
ad2@eng.cam.ac.uk

## Abstract

In this paper, we propose a full Bayesian model for neural networks. This model treats the model dimension (number of neurons), model parameters, regularisation parameters and noise parameters as random variables that need to be estimated. We then propose a reversible jump Markov chain Monte Carlo (MCMC) method to perform the necessary computations. We find that the results are not only better than the previously reported ones, but also appear to be robust with respect to the prior specification. Moreover, we present a geometric convergence theorem for the algorithm.

## 1 Introduction

In the early nineties, Buntine and Weigend (1991) and Mackay (1992) showed that a principled Bayesian learning approach to neural networks can lead to many improvements [1,2]. In particular, Mackay showed that by approximating the distributions of the weights with Gaussians and adopting smoothing priors, it is possible to obtain estimates of the weights and output variances and to automatically set the regularisation coefficients. Neal (1996) cast the net much further by introducing advanced Bayesian simulation methods, specifically the hybrid Monte Carlo method, into the analysis of neural networks [3]. Bayesian sequential Monte Carlo methods have also been shown to provide good training results, especially in time-varying scenarios [4]. More recently, Rios Insua and Müller (1998) and Holmes and Mallick (1998) have addressed the issue of selecting the number of hidden neurons with growing and pruning algorithms from a Bayesian perspective [5,6]. In particular, they apply the reversible jump Markov Chain Monte Carlo (MCMC) algorithm of Green [7] to feed-forward sigmoidal networks and radial basis function (RBF) networks to obtain joint estimates of the number of neurons and weights.

We also apply the reversible jump MCMC simulation algorithm to RBF networks so as to compute the joint posterior distribution of the radial basis parameters and the number of basis functions. However, we advance this area of research in two important directions. Firstly, we propose a full hierarchical prior for RBF networks. That

is, we adopt a full Bayesian model, which accounts for model order uncertainty and regularisation, and show that the results appear to be robust with respect to the prior specification. Secondly, we present a geometric convergence theorem for the algorithm. The complexity of the problem does not allow for a comprehensive discussion in this short paper. We have, therefore, focused on describing our objectives, the Bayesian model, convergence theorem and results. Readers are encouraged to consult our technical report for further results and implementation details [8][1].

## 2 Problem statement

Many physical processes may be described by the following nonlinear, multivariate input-output mapping:

$$\mathbf{y}_t = \mathbf{f}(\mathbf{x}_t) + \mathbf{n}_t \tag{1}$$

where $\mathbf{x}_t \in \mathbb{R}^d$ corresponds to a group of input variables, $\mathbf{y}_t \in \mathbb{R}^c$ to the target variables, $\mathbf{n}_t \in \mathbb{R}^c$ to an unknown noise process and $t = \{1, 2, \cdots\}$ is an index variable over the data. In this context, the learning problem involves computing an approximation to the function $\mathbf{f}$ and estimating the characteristics of the noise process given a set of $N$ input-output observations: $\mathcal{O} = \{\mathbf{x}_1, \mathbf{x}_2, \cdots, \mathbf{x}_N, \mathbf{y}_1, \mathbf{y}_2, \cdots, \mathbf{y}_N\}$ Typical examples include regression, where $\mathbf{y}_{1:N,1:c}$[2] is continuous; classification, where $\mathbf{y}$ corresponds to a group of classes and nonlinear dynamical system identification, where the inputs and targets correspond to several delayed versions of the signals under consideration.

We adopt the approximation scheme of Holmes and Mallick (1998), consisting of a mixture of $k$ RBFs and a linear regression term. Yet, the work can be easily extended to other regression models. More precisely, our model $\mathcal{M}$ is:

$$\mathcal{M}_0 : \qquad \mathbf{y}_t = b + \beta' \mathbf{x}_t + \mathbf{n}_t \qquad\qquad k = 0$$

$$\mathcal{M}_k : \quad \mathbf{y}_t = \sum_{j=1}^{k} \mathbf{a}_j \phi(\|\mathbf{x}_t - \boldsymbol{\mu}_j\|) + b + \beta' \mathbf{x}_t + \mathbf{n}_t \quad k \geq 1 \tag{2}$$

where $\| \cdot \|$ denotes a distance metric (usually Euclidean or Mahalanobis), $\boldsymbol{\mu}_j \in \mathbb{R}^d$ denotes the $j$-th RBF centre for a model with $k$ RBFs, $\mathbf{a}_j \in \mathbb{R}^c$ the $j$-th RBF amplitude and $b \in \mathbb{R}^c$ and $\beta \in \mathbb{R}^d \times \mathbb{R}^c$ the linear regression parameters. The noise sequence $\mathbf{n}_t \in \mathbb{R}^c$ is assumed to be zero-mean white Gaussian. It is important to mention that although we have not explicitly indicated the dependency of $b$, $\beta$ and $\mathbf{n}_t$ on $k$, these parameters are indeed affected by the value of $k$. For convenience, we express our approximation model in vector-matrix form:

$$
\begin{bmatrix} y_{1,1} \cdots y_{1,c} \\ y_{2,1} \cdots y_{2,c} \\ \vdots \\ y_{N,1} \cdots y_{N,c} \end{bmatrix}
=
\begin{bmatrix}
1 & x_{1,1} \cdots x_{1,d} & \phi(\mathbf{x}_1, \boldsymbol{\mu}_1) \cdots \phi(\mathbf{x}_1, \boldsymbol{\mu}_k) \\
1 & x_{2,1} \cdots x_{2,d} & \phi(\mathbf{x}_2, \boldsymbol{\mu}_1) \cdots \phi(\mathbf{x}_2, \boldsymbol{\mu}_k) \\
\vdots & \vdots & \vdots \\
1 & x_{N,1} \cdots x_{N,d} & \phi(\mathbf{x}_N, \boldsymbol{\mu}_1) \cdots \phi(\mathbf{x}_N, \boldsymbol{\mu}_k)
\end{bmatrix}
\begin{bmatrix}
b_1 \cdots b_c \\
\beta_{1,1} \cdots \beta_{1,c} \\
\vdots \\
\beta_{d,1} \cdots \beta_{d,c} \\
a_{1,1} \cdots a_{1,c} \\
\vdots \\
a_{k,1} \cdots a_{k,c}
\end{bmatrix}
+ \mathbf{n}_{1:N}
$$

where the noise process is assumed to be normally distributed $\mathbf{n}_t \sim \mathcal{N}(0, \sigma_i^2)$ for $i = 1, \ldots, c$. In shorter notation, we have:

$$\mathbf{y} = \mathbf{D}(\boldsymbol{\mu}_{1:k,1:d}, \mathbf{x}_{1:N,1:d})\boldsymbol{\alpha}_{1:1+d+k,1:c} + \mathbf{n}_t \qquad (3)$$

We assume here that the number $k$ of RBFs and their parameters $\boldsymbol{\theta} \triangleq \{\boldsymbol{\alpha}_{1:m,1:c}, \boldsymbol{\mu}_{1:k,1:d}, \sigma_{1:c}^2\}$, with $m = 1 + d + k$, are unknown. Given the data set $\{\mathbf{x}, \mathbf{y}\}$, our objective is to estimate $k$ and $\boldsymbol{\theta} \in \boldsymbol{\Theta}_k$.

## 3 Bayesian model and aims

We follow a Bayesian approach where the unknowns $k$ and $\boldsymbol{\theta}$ are regarded as being drawn from appropriate prior distributions. These priors reflect our degree of belief on the relevant values of these quantities [9]. Furthermore, we adopt a hierarchical prior structure that enables us to treat the priors' parameters (hyper-parameters) as random variables drawn from suitable distributions (hyper-priors). That is, instead of fixing the hyper-parameters arbitrarily, we acknowledge that there is an inherent uncertainty in what we think their values should be. By devising probabilistic models that deal with this uncertainty, we are able to implement estimation techniques that are robust to the specification of the hyper-priors.

The overall parameter space $\boldsymbol{\Theta} \times \boldsymbol{\Psi}$ can be written as a finite union of subspaces $\boldsymbol{\Theta} \times \boldsymbol{\Psi} = \left(\cup_{k=0}^{k_{\max}}\{k\} \times \boldsymbol{\Theta}_k\right) \times \boldsymbol{\Psi}$ where $\boldsymbol{\Theta}_0 \triangleq (\mathbb{R}^{d+1})^c \times (\mathbb{R}^+)^c$ and $\boldsymbol{\Theta}_k \triangleq (\mathbb{R}^{d+1+k})^c \times (\mathbb{R}^+)^c \times \boldsymbol{\Omega}_k$ for $k \in \{1, \ldots, k_{\max}\}$. That is, $\boldsymbol{\alpha} \in (\mathbb{R}^{d+1+k})^c$, $\boldsymbol{\sigma} \in (\mathbb{R}^+)^c$ and $\boldsymbol{\mu} \in \boldsymbol{\Omega}_k$. The hyper-parameter space $\boldsymbol{\Psi} \triangleq (\mathbb{R}^+)^{c+1}$, with elements $\boldsymbol{\psi} \triangleq \{\Lambda, \delta^2\}$, will be discussed at the end of this section. The space of the radial basis centres $\boldsymbol{\Omega}_k$ is defined as a compact set including the input data: $\boldsymbol{\Omega}_k \triangleq \{\boldsymbol{\mu}; \boldsymbol{\mu}_{1:k,i} \in [\min(\mathbf{x}_{1:N,i}) - \iota\Xi_i, \max(\mathbf{x}_{1:N,i}) + \iota\Xi_i]^k$ for $i = 1, \ldots, d$ with $\boldsymbol{\mu}_{j,i} \neq \boldsymbol{\mu}_{l,i}$ for $j \neq l\}$. $\Xi_i = \|\max(\mathbf{x}_{1:N,i}) - \min(\mathbf{x}_{1:N,i})\|$ denotes the Euclidean distance for the $i$-th dimension of the input and $\iota$ is a user specified parameter that we only need to consider if we wish to place basis functions outside the region where the input data lie. That is, we allow $\boldsymbol{\Omega}_k$ to include the space of the input data and extend it by a factor which is proportional to the spread of the input data. The hyper-volume of this space is: $\Im^k \triangleq \left(\prod_{i=1}^d (1 + 2\iota)\Xi_i\right)^k$.

The maximum number of basis functions is defined as $k_{\max} \triangleq (N - (d + 1))$ We also define $\boldsymbol{\Omega} \triangleq \cup_{k=0}^{k_{\max}}\{k\} \times \boldsymbol{\Omega}_k$ with $\boldsymbol{\Omega}_0 \triangleq \emptyset$. Under the assumption of independent outputs given $(k, \boldsymbol{\theta})$, the likelihood $p(\mathbf{y}|k, \boldsymbol{\theta}, \boldsymbol{\psi}, \mathbf{x})$ for the approximation model described in the previous section is:

$$\prod_{i=1}^c (2\pi\sigma_i^2)^{-N/2} \exp\left(-\frac{1}{2\sigma_i^2}\left(\mathbf{y}_{1:N,i} - \mathbf{D}(\boldsymbol{\mu}_{1:k}, \mathbf{x})\boldsymbol{\alpha}_{1:m,i}\right)'\left(\mathbf{y}_{1:N,i} - \mathbf{D}(\boldsymbol{\mu}_{1:k}, \mathbf{x})\boldsymbol{\alpha}_{1:m,i}\right)\right)$$

We assume the following structure for the prior distribution:

$$p(k, \boldsymbol{\theta}, \boldsymbol{\psi}) = p(\boldsymbol{\alpha}_{1:m}|k, \sigma^2, \delta^2)p(\boldsymbol{\mu}_{1:k}|k)p(k|\Lambda)p(\sigma^2)p(\Lambda)p(\delta^2)$$

where the scale parameters $\sigma_i^2$, are assumed to be independent of the hyper-parameters (i.e. $p(\sigma^2|\Lambda, \delta^2) = p(\sigma^2)$), independent of each other ($p(\sigma^2) = \prod_{i=1}^c p(\sigma_i^2)$) and distributed according to conjugate inverse-Gamma prior distributions: $\sigma_i^2 \sim \mathcal{IG}\left(\frac{v_0}{2}, \frac{\gamma_0}{2}\right)$. When $v_0 = 0$ and $\gamma_0 = 0$, we obtain Jeffreys' uninformative prior [9]. For a given $\sigma^2$, the prior distribution $p(k, \boldsymbol{\alpha}_{1:m}, \boldsymbol{\mu}_{1:k}|\sigma^2, \Lambda, \delta^2)$ is:

$$\left[\prod_{i=1}^c |2\pi\sigma_i^2\delta_i^2\mathbf{I}_m|^{-1/2} \exp\left(-\frac{1}{2\sigma_i^2\delta_i^2}\boldsymbol{\alpha}_{1:m,i}'\boldsymbol{\alpha}_{1:m,i}\right)\right]\left[\frac{\mathbb{I}_{\boldsymbol{\Omega}}(k, \boldsymbol{\mu}_{1:k})}{\Im^k}\right]\left[\frac{\Lambda^k/k!}{\sum_{j=0}^{k_{\max}}\Lambda^j/j!}\right]$$

where $\mathbf{I}_m$ denotes the identity matrix of size $m \times m$ and $\mathbb{I}_\Omega(k, \boldsymbol{\mu}_{1:k})$ is the indicator function of the set $\Omega$ (1 if $(k, \boldsymbol{\mu}_{1:k}) \in \Omega$, 0 otherwise).

The prior model order distribution $p(k|\Lambda)$ is a truncated Poisson distribution. Conditional upon $k$, the RBF centres are uniformly distributed. Finally, conditional upon $(k, \boldsymbol{\mu}_{1:k})$, the coefficients $\boldsymbol{\alpha}_{1:m,i}$ are assumed to be zero-mean Gaussian with variance $\delta_i^2 \sigma_i^2$. The hyper-parameters $\boldsymbol{\delta}^2 \in (\mathbb{R}^+)^c$ and $\Lambda \in \mathbb{R}^+$ can be respectively interpreted as the expected signal to noise ratios and the expected number of radial basis. We assume that they are independent of each other, i.e. $p(\Lambda, \boldsymbol{\delta}^2) = p(\Lambda)p(\boldsymbol{\delta}^2)$. Moreover, $p(\boldsymbol{\delta}^2) = \prod_{i=1}^c p(\delta_i^2)$. As $\boldsymbol{\delta}^2$ is a scale parameter, we ascribe a vague conjugate prior density to it: $\delta_i^2 \sim \mathcal{IG}(\alpha_{\delta^2}, \beta_{\delta^2})$ for $i = 1, \ldots, c$, with $\alpha_{\delta^2} = 2$ and $\beta_{\delta^2} > 0$. The variance of this hyper-prior with $\alpha_{\delta^2} = 2$ is infinite. We apply the same method to $\Lambda$ by setting an uninformative conjugate prior [9]: $\Lambda \sim \mathcal{G}a(1/2 + \varepsilon_1, \varepsilon_2)$ $(\varepsilon_i \ll 1\ i = 1, 2)$.

### 3.1 Estimation and inference aims

The Bayesian inference of $k$, $\boldsymbol{\theta}$ and $\boldsymbol{\psi}$ is based on the joint posterior distribution $p(k, \boldsymbol{\theta}, \boldsymbol{\psi}|\mathbf{x}, \mathbf{y})$ obtained from Bayes' theorem. Our aim is to estimate this joint distribution from which, by standard probability marginalisation and transformation techniques, one can "theoretically" obtain all posterior features of interest. We propose here to use the reversible jump MCMC method to perform the necessary computations, see [8] for details. MCMC techniques were introduced in the mid 1950's in statistical physics and started appearing in the fields of applied statistics, signal processing and neural networks in the 1980's and 1990's [3,5,6,10,11]. The key idea is to build an ergodic Markov chain $(k^{(i)}, \boldsymbol{\theta}^{(i)}, \boldsymbol{\psi}^{(i)})_{i \in \mathbb{N}}$ whose equilibrium distribution is the desired posterior distribution. Under weak additional assumptions, the $P \gg 1$ samples generated by the Markov chain are asymptotically distributed according to the posterior distribution and thus allow easy evaluation of all posterior features of interest. For example:

$$\widehat{p}(k = j|\mathbf{x}, \mathbf{y}) = \frac{1}{P}\sum_{i=1}^P \mathbb{I}_{\{j\}}(k^{(i)}) \quad \text{and} \quad \widehat{\mathbb{E}}(\boldsymbol{\theta}|k = j, \mathbf{x}, \mathbf{y}) = \frac{\sum_{i=1}^P \boldsymbol{\theta}^{(i)}\mathbb{I}_{\{j\}}(k^{(i)})}{\sum_{i=1}^P \mathbb{I}_{\{j\}}(k^{(i)})} \quad (4)$$

In addition, we can obtain predictions, such as:

$$\widehat{\mathbb{E}}(\mathbf{y}_{N+1}|\mathbf{x}_{1:N+1}, \mathbf{y}_{1:N}) = \frac{1}{P}\sum_{i=1}^P \mathbf{D}(\boldsymbol{\mu}_{1:k}^{(i)}, \mathbf{x}_{N+1})\boldsymbol{\alpha}_{1:m}^{(i)} \quad (5)$$

### 3.2 Integration of the nuisance parameters

According to Bayes theorem, we can obtain the posterior distribution as follows:

$$p(k, \boldsymbol{\theta}, \boldsymbol{\psi}|\mathbf{x}, \mathbf{y}) \propto p(\mathbf{y}|k, \boldsymbol{\theta}, \boldsymbol{\psi}, \mathbf{x})p(k, \boldsymbol{\theta}, \boldsymbol{\psi})$$

In our case, we can integrate with respect to $\boldsymbol{\alpha}_{1:m}$ (Gaussian distribution) and with respect to $\sigma_i^2$ (inverse Gamma distribution) to obtain the following expression for the posterior:

$$p(k, \boldsymbol{\mu}_{1:k}, \Lambda, \boldsymbol{\delta}^2|\mathbf{x}, \mathbf{y}) \propto \left[\prod_{i=1}^c (\delta_i^2)^{-m/2}|\mathbf{M}_{i,k}|^{1/2}\left(\frac{\gamma_0 + \mathbf{y}_{1:N,i}'\mathbf{P}_{i,k}\mathbf{y}_{1:N,i}}{2}\right)^{(-\frac{N+v_0}{2})}\right] \times$$

$$\left[\frac{\mathbb{I}_\Omega(k, \boldsymbol{\mu}_k)}{\mathfrak{S}^k}\right]\left[\frac{\Lambda^k/k!}{\sum_{j=0}^{k_{\max}} \Lambda^j/j!}\right]\left[\prod_{i=1}^c (\delta_i^2)^{-(\alpha_{\delta^2}+1)}\exp\left(-\frac{\beta_{\delta^2}}{\delta_i^2}\right)\right]\left[(\Lambda)^{(\varepsilon_1-1/2)}\exp\left(-\varepsilon_2\Lambda\right)\right]$$

$$(6)$$

It is worth noticing that the posterior distribution is highly non-linear in the RBF centres $\mu_k$ and that an expression of $p(k|\mathbf{x}, \mathbf{y})$ cannot be obtained in closed-form.

## 4   Geometric convergence theorem

It is easy to prove that the reversible jump MCMC algorithm applied to our model converges, that is, that the Markov chain $\left(k^{(i)}, \mu_{1:k}^{(i)}, \Lambda^{(i)}, \delta^{2(i)}\right)_{i \in \mathbb{N}}$ is ergodic. We present here a stronger result, namely that $\left(k^{(i)}, \mu_{1:k}^{(i)}, \Lambda^{(i)}, \delta^{2(i)}\right)_{i \in \mathbb{N}}$ converges to the required posterior distribution at a geometric rate:

**Theorem 1** *Let* $\left(k^{(i)}, \mu_{1:k}^{(i)}, \Lambda^{(i)}, \delta^{2(i)}\right)_{i \in \mathbb{N}}$ *be the Markov chain whose transition kernel has been described in Section 3. This Markov chain converges to the probability distribution* $p\left(k, \mu_{1:k}, \Lambda, \delta^2 \,|\, \mathbf{x}, \mathbf{y}\right)$. *Furthermore this convergence occurs at a geometric rate, that is, for almost every initial point* $\left(k^{(0)}, \mu_{1:k}^{(0)}, \Lambda^{(0)}, \delta^{2(0)}\right) \in \Omega \times \Psi$ *there exists a function of the initial states* $C_0 > 0$ *and a constant and* $\rho \in [0, 1)$ *such that*

$$\left\| p^{(i)}\left(k, \mu_{1:k}, \Lambda, \delta^2\right) - p\left(k, \mu_{1:k}, \Lambda, \delta^2 \,\big|\, \mathbf{x}, \mathbf{y}\right) \right\|_{TV} \leq C_0 \rho^{\lfloor i/k_{\max} \rfloor} \tag{7}$$

*where* $p^{(i)}\left(k, \mu_{1:k}, \Lambda, \delta^2\right)$ *is the distribution of* $\left(k^{(i)}, \mu_{1:k}^{(i)}, \Lambda^{(i)}, \delta^{2(i)}\right)$ *and* $\|\cdot\|_{TV}$ *is the total variation norm* [11]. **Proof.** *See* [8] ∎

**Corollary 1** *If for each iteration i one samples the nuisance parameters* $(\alpha_{1:m}, \sigma_k^2)$ *then the distribution of the series* $(k^{(i)}, \alpha_{1:m}^{(i)}, \mu_{1:k}^{(i)}, \sigma_k^{2(i)}, \Lambda^{(i)}, \delta^{2(i)})_{i \in \mathbb{N}}$ *converges geometrically towards* $p(k, \alpha_{1:m}, \mu_{1:k}, \sigma_k^2, \Lambda, \delta^2 | \mathbf{x}, \mathbf{y})$ *at the same rate* $\rho$.

## 5   Demonstration: robot arm data

This data is often used as a benchmark to compare learning algorithms[3]. It involves implementing a model to map the joint angle of a robot arm $(x_1, x_2)$ to the position of the end of the arm $(y_1, y_2)$. The data were generated from the following model:

$$\begin{aligned} y_1 &= 2.0\cos(x_1) + 1.3\cos(x_1 + x_2) + \epsilon_1 \\ y_2 &= 2.0\sin(x_1) + 1.3\sin(x_1 + x_2) + \epsilon_2 \end{aligned}$$

where $\epsilon_i \sim \mathcal{N}(0, \sigma^2)$; $\sigma = 0.05$. We use the first 200 observations of the data set to train our models and the last 200 observations to test them. In the simulations, we chose to use cubic basis functions. Figure 1 shows the 3D plots of the training data and the contours of the training and test data. The contour plots also include the typical approximations that were obtained using the algorithm. We chose uninformative priors for all the parameters and hyper-parameters (Table 1). To demonstrate the robustness of our algorithm, we chose different values for $\beta_{\delta^2}$ (the only critical hyper-parameter as it quantifies the mean of the spread $\delta$ of $\alpha_k$). The obtained mean square errors and probabilities for $\delta_1$, $\delta_2$, $\sigma_{1,k}^2$, $\sigma_{2,k}^2$ and $k$, shown in Figure 2, clearly indicate that our algorithm is robust with respect to prior specification. Our mean square errors are of the same magnitude as the ones reported by other researchers [2,3,5,6]; slightly better (Not by more than 10%). Moreover, our algorithm leads to more parsimonious models than the ones previously reported.

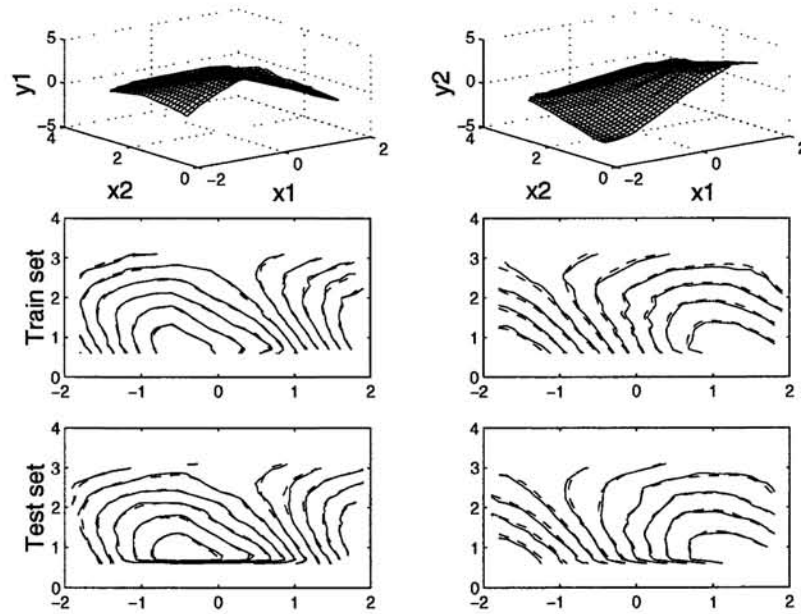

Figure 1: The top plots show the training data surfaces corresponding to each coordinate of the robot arm's position. The middle and bottom plots show the training and validation data [- -] and the respective RBF network mappings [—].

Table 1: Simulation parameters and mean square test errors.

| $\alpha_{\delta^2}$ | $\beta_{\delta^2}$ | $v_0$ | $\gamma_0$ | $\varepsilon_1$ | $\varepsilon_2$ | MS ERROR |
|---|---|---|---|---|---|---|
| 2 | 0.1 | 0 | 0 | 0.0001 | 0.0001 | 0.00505 |
| 2 | 10 | 0 | 0 | 0.0001 | 0.0001 | 0.00503 |
| 2 | 100 | 0 | 0 | 0.0001 | 0.0001 | 0.00502 |

## 6    Conclusions

We presented a general methodology for estimating, jointly, the noise variance, parameters and number of parameters of an RBF model. In adopting a Bayesian model and the reversible jump MCMC algorithm to perform the necessary integrations, we demonstrated that the method is very accurate. Contrary to previous reported results, our experiments indicate that our model is robust with respect to the specification of the prior. In addition, we obtained more parsimonious RBF networks and better approximation errors than the ones previously reported in the literature. There are many avenues for further research. These include estimating the type of basis functions, performing input variable selection, considering other noise models and extending the framework to sequential scenarios. A possible solution to the first problem can be formulated using the reversible jump MCMC framework. Variable selection schemes can also be implemented via the reversible jump MCMC algorithm. We are presently working on a sequential version of the algorithm that allows us to perform model selection in non-stationary environments.

## Footnotes

*Authorship based on alphabetical order.

[1]The software is available at http://www.cs.berkeley.edu/~jfgf.

[2]$\mathbf{y}_{1:N,1:c}$ is an $N$ by $c$ matrix, where $N$ is the number of data and $c$ the number of outputs. We adopt the notation $\mathbf{y}_{1:N,j} \triangleq (\mathbf{y}_{1,j}, \mathbf{y}_{2,j}, \ldots, \mathbf{y}_{N,j})'$ to denote all the observations corresponding to the $j$-th output ($j$-th column of $\mathbf{y}$). To simplify the notation, $\mathbf{y}_t$ is equivalent to $\mathbf{y}_{t,1:c}$. That is, if one index does not appear, it is implied that we are referring to all of its possible values. Similarly, $\mathbf{y}$ is equivalent to $\mathbf{y}_{1:N,1:c}$. We will favour the shorter notation and only adopt the longer notation to avoid ambiguities and emphasise certain dependencies.

[3]The robot arm data set can be found in David Mackay's home page: http://wol.ra.phy.cam.ac.uk/mackay/

## References

[1] Buntine, W.L. & Weigend, A.S. (1991) Bayesian back-propagation. *Complex Systems* **5**:603-643.

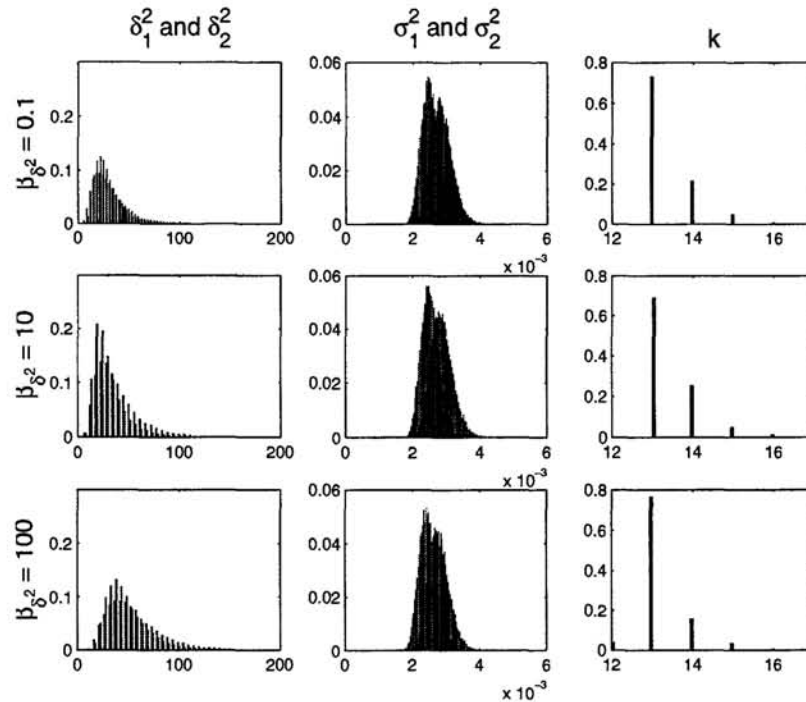

Figure 2: Histograms of smoothness constraints ($\delta_1$ and $\delta_2$), noise variances ($\sigma^2_{1,k}$ and $\sigma^2_{2,k}$) and model order ($k$) for the robot arm data using 3 different values for $\beta_{\delta^2}$. The plots confirm that the algorithm is robust to the setting of $\beta_{\delta^2}$.

[2] Mackay, D.J.C. (1992) A practical Bayesian framework for backpropagation networks. *Neural Computation* 4:448-472.

[3] Neal, R.M. (1996) *Bayesian Learning for Neural Networks*. New York: Lecture Notes in Statistics No. 118, Springer-Verlag.

[4] de Freitas, J.F.G., Niranjan, M., Gee, A.H. & Doucet, A. (1999) Sequential Monte Carlo methods to train neural network models. To appear in *Neural Computation*.

[5] Rios Insua, D. & Müller, P. (1998) Feedforward neural networks for nonparametric regression. *Technical report* 98-02. Institute of Statistics and Decision Sciences, Duke University, http://www.stat.duke.edu.

[6] Holmes, C.C. & Mallick, B.K. (1998) Bayesian radial basis functions of variable dimension. *Neural Computation* 10:1217-1233.

[7] Green, P.J. (1995) Reversible jump Markov chain Monte Carlo computation and Bayesian model determination. *Biometrika* 82:711-732.

[8] Andrieu, C., de Freitas, J.F.G. & Doucet, A. (1999) Robust full Bayesian learning for neural networks. *Technical report* CUED/F-INFENG/TR 343. Cambridge University, http://svr-www.eng.cam.ac.uk/.

[9] Bernardo, J.M. & Smith, A.F.M. (1994) *Bayesian Theory*. Chichester: Wiley Series in Applied Probability and Statistics.

[10] Besag, J., Green, P.J., Hidgon, D. & Mengersen, K. (1995) Bayesian computation and stochastic systems. *Statistical Science* 10:3-66.

[11] Tierney, L. (1994) Markov chains for exploring posterior distributions. *The Annals of Statistics*. 22(4):1701-1762.
